# Model Complexity, Goodness of Fit and Diminishing Returns

**Igor V. Cadez**
Information and Computer Science
University of California
Irvine, CA 92697-3425, U.S.A.

**Padhraic Smyth**
Information and Computer Science
University of California
Irvine, CA 92697-3425, U.S.A.

## Abstract

We investigate a general characteristic of the trade-off in learning problems between goodness-of-fit and model complexity. Specifically we characterize a general class of learning problems where the goodness-of-fit function can be shown to be convex within first-order as a function of model complexity. This general property of "diminishing returns" is illustrated on a number of real data sets and learning problems, including finite mixture modeling and multivariate linear regression.

## 1 Introduction, Motivation, and Related Work

Assume we have a data set $D = \{x_1, x_2, \ldots, x_n\}$, where the $x_i$ could be vectors, sequences, etc. We consider modeling the data set $D$ using models indexed by a complexity index $k$, $1 \leq k \leq k_{\max}$. For example, the models could be finite mixture probability density functions (PDFs) for vector $x_i$'s where model complexity is indexed by the number of components $k$ in the mixture. Alternatively, the modeling task could be to fit a conditional regression model $y = g(z_k) + e$, where now $y$ is one of the variables in the vector $x$ and $z$ is some subset of size $k$ of the remaining components in the $x$ vector.

Such learning tasks can typically be characterized by the existence of a *model* and a *loss function*. A fitted model of complexity $k$ is a function of the data points $D$ and depends on a specific set of fitted *parameters* $\theta$. The loss function (goodness-of-fit) is a functional of the model and maps each specific model to a scalar used to evaluate the model, e.g., likelihood for density estimation or sum-of-squares for regression.

Figure 1 illustrates a typical empirical curve for loss function versus complexity, for mixtures of Markov models fitted to a large data set of 900,000 sequences. The complexity $k$ is the number of Markov models being used in the mixture (see Cadez et al. (2000) for further details on the model and the data set). The empirical curve has a distinctly concave appearance, with large relative gains in fit for low complexity models and much more modest relative gains for high complexity models. A natural question is whether this concavity characteristic can be viewed as a general phenomenon in learning and under what assumptions on model classes and

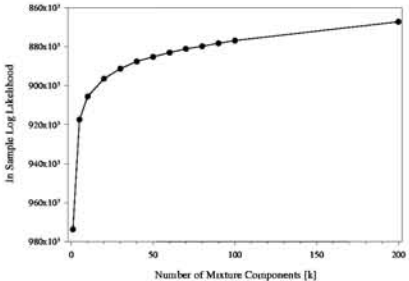

Figure 1: Log-likelihood scores for a Markov mixtures data set.

loss functions the concavity can be shown to hold. The goal of this paper is to illustrate that in fact it is a natural characteristic for a broad range of problems in mixture modeling and linear regression.

We note of course that for *generalization* that using goodness-of-fit alone will lead to the selection of the most complex model under consideration and will not in general select the model which generalizes best to new data. Nonetheless our primary focus of interest in this paper is how goodness-of-fit loss functions (such as likelihood and squared error, defined on the training data $D$) behave in general as a function of model complexity $k$. Our concavity results have a number of interesting implications. For example, for model selection methods which add a penalty term to the goodness-of-fit (e.g., BIC), the resulting score function as a function of model complexity will be unimodal as a function of complexity $k$ within first order.

Li and Barron (1999) have shown that for finite mixture models the expected value of the log-likelihood for any $k$ is bounded below by a function of the form $-C/k$ where $C$ is a constant which is independent of $k$. The results presented here are complementary in the sense that we show that the actual maximizing log-likelihood itself is concave to first-order as a function of $k$. Furthermore, we obtain a more general principle of "diminishing returns," including both finite mixtures and subset selection in regression.

## 2 Notation

We define $y = y(x)$ as a scalar function of $x$, namely a prediction at $x$. In linear regression $y = y(x)$ is a linear function of the components in $x$ while in density estimation $y = y(x)$ is the value of the density function at $x$. Although the goals of regression and density estimation are quite different, we can view them both as simply techniques for approximating an unknown true function for different values of $x$. We denote the prediction of a model of complexity $k$ as $y_k(x|\theta)$ where the subscript indicates the model complexity and $\theta$ is the associated set of fitted parameters. Since different choices of parameters in general yield different models, we will typically abbreviate the notation somewhat and use different letters for different parameterizations of the same functional form (i.e., the same complexity), e.g., we may use $y_k(x), g_k(x), h_k(x)$ to refer to models of complexity $k$ instead of specifying $y_k(x|\theta_1), y_k(x|\theta_2), y_k(x|\theta_3)$, etc. Furthermore, since all models under discussion are functions of $x$, we sometimes omit the explicit dependence on $x$ and use a compact notation $y_k, g_k, h_k$.

We focus on classes of models that can be characterized by more complex models having a *linear* dependence on simpler models within the class. More formally, any

model of complexity $k$ can be decomposed as:

$$y_k = \alpha_1 g_1 + \alpha_2 h_1 + \ldots + \alpha_k w_1. \tag{1}$$

In PDF mixture modeling we have $y_k = p(x)$ and each model $g_1, h_1, \ldots, z_1$ is a basis PDF (e.g., a single Gaussian) but with different parameters. In multivariate linear regression each model $g_1, h_1, \ldots, w_1$ represents a regression on a single variable, e.g., $g_1(x)$ above is $g_1(x) = \gamma_p x_p$ where $x_p$ is the $p$-th variable in the set and $\gamma_p$ is the corresponding coefficient one would obtain if regressing on $x_p$ alone. One of the $g_1, h_1, \ldots, w_1$ can be a dummy constant variable to account for the intercept term. Note that the total parameters for the model $y_k$ in both cases can be viewed as consisting of both the mixing proportions (the $\alpha$'s) and the parameters for each individual component model.

The loss function is a functional on models and we write it as $E(y_k)$. For simplicity, we use the notation $E_k^*$ to specify the value of the loss function for the *best $k$-component* model. This way, $E_k^* \leq E(y_k)$ for any model $y_k$[1]. For example, the loss function in PDF mixture modeling is the negative log likelihood. In linear regression we use empirical mean squared error (MSE) as the loss function. The loss functions of general interest in this context are those that decompose into a sum of functions over data points in the data set $D$ (equivalently an independence assumption in a likelihood framework), i.e.,

$$E(y_k) = \sum_{i=1}^{n} f(y_k(x_i)) \tag{2}$$

For example, in PDF mixture modeling $f(y_k) = -\ln y_k$, while in regression modeling $f(y_k) = (y - y_k)^2$ where $y$ is a known target value.

## 3 Necessary Conditions on Models and Loss Functions

We consider models that satisfy several conditions that are commonly met in real data analysis applications and are satisfied by both PDF mixture models and linear regression models:

1. As $k$ increases we have a nested model class, i.e., each model of complexity $k$ contains each model of complexity $k' < k$ as a special case (i.e., it reduces to a simpler model for a special choice of the parameters).

2. Any two models of complexities $k_1$ and $k_2$ can be combined as a weighted sum in any proportion to yield a valid model of complexity $k = k_1 + k_2$.

3. Each model of complexity $k = k_1 + k_2$ can be decomposed into a weighted sum of two valid models of complexities $k_1$ and $k_2$ respectively for each valid choice of $k_1$ and $k_2$.

The first condition guarantees that the loss function is a non-increasing function of $k$ for optimal models of complexity $k$ (in sense of minimizing the loss function $E$), the second condition prevents artificial correlation between the component models, while the third condition guarantees that all components are of equal expressive power. As an example, the standard Gaussian mixture model satisfies all three properties whether the covariance matrices are unconstrained or individually constrained. As a counter-example, a Gaussian mixture model where the covariance matrices are constrained to be equal across all components does not satisfy the second property.

# 4    Theoretical Results on Loss Function Convexity

We formulate and prove the following theorem:

*Theorem 1:* In a learning problem that satisfies the properties from Section 3, the loss function is first order convex in model complexity $k$, meaning that $E_{k+1}^* - 2E_k^* + E_{k-1}^* \geq 0$ within first order (as defined in the proof). The quantities $E_k^*$ and $E_{k\pm1}^*$ are the values of the loss function for the best $k$ and $k \pm 1$-component models.

*Proof:* In the first part of the proof we analyze a general difference of loss functions and write it in a convenient form. Consider two arbitrary models, $g$ and $h$ and the corresponding loss functions $E(g)$ and $E(h)$ ($g$ and $h$ need not have the same complexity). The difference in loss functions can be expressed as:

$$
\begin{aligned}
E(g) - E(h) &= \sum_{i=1}^{n} \left\{ f\left[g(x_i)\right] - f\left[h(x_i)\right] \right\} \\
&= \sum_{i=1}^{n} \left\{ f\left[h(x_i)(1 + \delta_{g,h}(x_i))\right] - f\left[h(x_i)\right] \right\} \\
&= \alpha \sum_{i=1}^{n} h(x_i) f'\left(h(x_i)\right) \delta_{g,h}(x_i).
\end{aligned} \tag{3}
$$

where the last equation comes from a first order Taylor series expansion around each $\delta_{g,h}(x_i) = 0$, $\alpha$ is an unknown constant of proportionality (to make the equation exact) and

$$
\delta_{g,h}(x) \doteq \frac{g(x) - h(x)}{h(x)} \tag{4}
$$

represents the *relative difference* in models $g$ and $h$ at point $x$. For example, Equation 3 reduces to a first order Taylor series approximation for $\alpha = 1$. If $f(y)$ is a convex function we also have:

$$
E(g) - E(h) \geq \sum_{i=1}^{n} h(x_i) f'(h(x_i)) \delta_{g,h}(x_i). \tag{5}
$$

since the remainder in the Taylor series expansion $R_2 = 1/2 f''(h(1 + \Theta\delta))\delta^2 \geq 0$.

In the second part of the proof we use Equation 5 to derive an appropriate condition on loss functions. Consider the best $k$ and $k \pm 1$-component models and the appropriate difference of the corresponding loss functions $E_{k+1}^* - 2E_k^* + E_{k-1}^*$, which we can write using the notation from Equation 3 and Equation 5 (since we consider convex functions $f(y) = -\ln y$ for PDF modeling and $f(y) = (y - y_i)^2$ for best subset regression) as:

$$
\begin{aligned}
E_{k+1}^* &- 2E_k^* + E_{k-1}^* = \\
&= \sum_{i=1}^{n} \left[ f(y_{k+1}^*(x_i)) - f(y_k^*(x_i)) \right] + \sum_{i=1}^{n} \left[ f(y_{k-1}^*(x_i)) - f(y_k^*(x_i)) \right] \\
&\geq \sum_{i=1}^{n} y_k^*(x_i) f'(y_k^*(x_i)) \delta_{y_{k+1}^*,y_k^*}(x_i) + \sum_{i=1}^{n} y_k^*(x_i) f'(y_k^*(x_i)) \delta_{y_{k-1}^*,y_k^*}(x_i) \\
&= \sum_{i=1}^{n} y_k^*(x_i) f'(y_k^*(x_i)) \left[ \delta_{y_{k+1}^*,y_k^*}(x_i) + \delta_{y_{k-1}^*,y_k^*}(x_i) \right].
\end{aligned} \tag{6}
$$

According to the requirements on models in Section 3, the best $k+1$-component model can be decomposed as

$$y_{k+1}^* = (1-\epsilon)g_k + \epsilon g_1,$$

where $g_k$ is a $k$-component model and $g_1$ is a 1-component model. Similarly, an artificial model can be constructed from the best $k-1$-component model:

$$\xi_k = (1-\epsilon)y_{k-1}^* + \epsilon g_1.$$

Upon subtracting $y_k^*$ from each of the equations and dividing by $y_k^*$, using notation from Equation 4, we get:

$$\delta_{y_{k+1}^*,y_k^*} = (1-\epsilon)\delta_{g_k,y_k^*} + \epsilon\delta_{g_1,y_k^*}$$
$$\delta_{\xi_k,y_k^*} = (1-\epsilon)\delta_{y_{k-1}^*,y_k^*} + \epsilon\delta_{g_1,y_k^*},$$

which upon subtraction and rearrangement of terms yields:

$$\delta_{y_{k+1}^*,y_k^*} + \delta_{y_{k-1}^*,y_k^*} = (1-\epsilon)\delta_{g_k,y_k^*} + \delta_{\xi_k,y_k^*} + \epsilon\delta_{y_{k-1}^*,y_k^*}. \tag{7}$$

If we evaluate this equation at each of the data points $x_i$ and substitute the result back into equation 6 we get:

$$E_{k+1}^* - 2E_k^* + E_{k-1}^* \geq$$
$$\sum_{i=1}^{n} y_k^*(x_i)f'(y_k^*(x_i))\left[(1-\epsilon)\delta_{g_k,y_k^*}(x_i) + \delta_{\xi_k,y_k^*}(x_i) + \epsilon\delta_{y_{k-1}^*,y_k^*}(x_i)\right]. \tag{8}$$

In the third part of the proof we analyze each of the terms in Equation 8 using Equation 3. Consider the first term:

$$\Delta_{g_k,y_k^*} = \sum_{i=1}^{n} y_k^*(x_i)f'(y_k^*(x_i))\delta_{g_k,y_k^*}(x_i) \tag{9}$$

that depends on a relative difference of models $g_k$ and $y_k^*$ at each of the data points $x_i$. According to Equation 3, for small $\delta_{g_k,y_k^*}(x_i)$ (which is presumably true), we can set $\alpha \approx 1$ to get a first order Taylor expansion. Since $y_k^*$ is the best $k$-component model, we have $E(g_k) \geq E(y_k^*) = E_k^*$ and consequently

$$E(g_k) - E(y_k^*) = \alpha\Delta_{g_k,y_k^*} \approx \Delta_{g_k,y_k^*} \geq 0 \tag{10}$$

Note that in order to have the last inequality hold, we do not require that $\alpha \approx 1$, but only that

$$\alpha \geq 0 \tag{11}$$

which is a weaker condition that we refer to as the *first order approximation*. In other words, we only require that the *sign* is preserved when making Taylor expansion while the actual value need not be very accurate. Similarly, each of the three terms on the right hand side of Equation 8 is first order positive since $E(y_k^*) \leq E(g_k), E(\xi_k), E(y_{k-1}^*)$. This shows that

$$E_{k+1}^* - 2E_k^* + E_{k-1}^* \geq 0$$

within first order, concluding the proof.

## 5 Convexity in Common Learning Problems

In this section we specialize Theorem 1 to several well-known learning situations. Each proof consists of merely selecting the appropriate loss function $E(y)$ and model family $y$.

## 5.1 Concavity of Mixture Model Log-Likelihoods

*Theorem 2:* In mixture model learning, using log-likelihood as the loss function and using unconstrained mixture components, the in-sample log likelihood is a first-order concave function of the complexity $k$.

*Proof:* By using $f(y) = -\ln y$ in Theorem 1 the loss function $E(y)$ becomes the negative of the in-sample log likelihood, hence it is a first-order convex function of complexity $k$, i.e., the log likelihood is first-order concave.

*Corollary 1:* If a linear or convex penalty term in $k$ is subtracted from the in-sample log likelihood in Theorem 2, using the mixture models as defined in Theorem 2, then the penalized likelihood can have at most one maximum to within first order. The BIC criterion satisfies this criterion for example.

## 5.2 Convexity of Mean-Square-Error for Subset Selection in Linear Regression

*Theorem 3:* In linear regression learning where $y_k$ represents the best linear regression defined over all possible subsets of $k$ regression variables, the mean squared error (MSE) is first-order convex as a function of the complexity $k$.

*Proof:* We use $f(y_k(x_i)) = (y_i - y_k(x_i))^2$ which is a convex function of $y_k$. The corresponding loss function $E(y_k)$ becomes the mean-square-error and is first-order convex as a function of the complexity $k$ by the proof of Theorem 1.

*Corollary 2:* If a concave or linear penalty term in $k$ is added to the mean squared error as defined in Theorem 3, then the resulting penalized mean-square-error can have at most one minimum to within first order. Such penalty terms include Mallow's $C_p$ criterion, AIC, BIC, predicted squared error, etc., (e.g., see Bishop (1995)).

# 6 Experimental Results

In this section we demonstrate empirical evidence of the approximate concavity property on three different data sets with model families and loss functions which satisfy the assumptions stated earlier:

1. *Mixtures of Gaussians*: 3962 data points in 2 dimensions, representing the first two principal components of historical geopotential data from upper-atmosphere data records, were fit with a mixture of $k$ Gaussian components, $k$ varying from 1 to 20 (see Smyth, Ide, and Ghil (1999) for more discussion of this data). Figure 2(a) illustrates that the log-likelihood is approximately concave as a function of $k$. Note that it is not completely concave. This could be a result of either local maxima in the fitting process (the maximum likelihood solutions in the interior of parameter space were selected as the best obtained by EM from 10 different randomly chosen initial conditions), or may indicate that concavity cannot be proven beyond a first-order characterization in the general case.

2. *Mixtures of Markov Chains*: Page-request sequences logged at the msnbc.com Web site over a 24-hour period from over 900,000 individuals were fit with mixtures of first-order Markov chains (see Cadez et al. (2000) for further details). Figure 1 again clearly shows a concave characteristic for the log-likelihood as a function of $k$, the number of Markov components in the model.

3. *Subset Selection in Linear Regression*: Autoregressive (AR) linear models were fit (closed form solutions for the optimal model parameters) to a monthly financial time series with 307 observations, for all possible combinations of lags (all possible

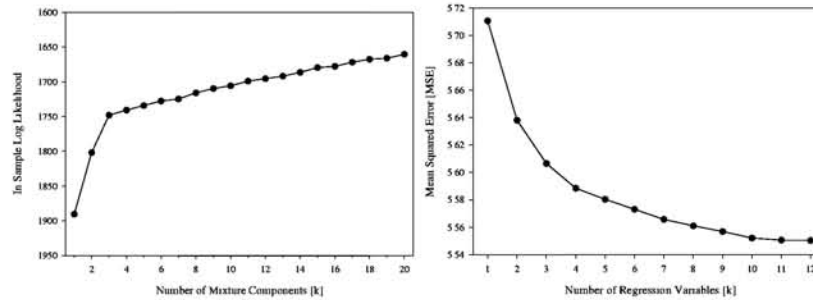

Figure 2: (a) In-sample log-likelihood for mixture modeling of the atmospheric data set, (b) mean-squared error for regression using the financial data set.

subsets) from order $k = 1$ to order $k = 12$. For example, the $k = 1$ model represents the best model with a single predictor from the previous 12 months, not necessarily the AR(1) model. Again the goodness-of-fit curve is almost convex in $k$ (Figure 2(b)), except at $k = 9$ where there is a slight non-concavity: this could again be either a numerical estimation effect or a fundamental characteristic indicating that concavity is only true to first-order.

## 7 Discussion and Conclusions

Space does not permit a full discussion of the various implications of the results derived here. The main implication is that for at least two common learning scenarios the maximizing/minimizing value of the loss function is strongly constrained as model complexity is varied. Thus, for example, when performing model selection using penalized goodness-of-fit (as in the Corollaries above) variants of binary search may be quite useful in problems where $k$ is very large (in the mixtures of Markov chains above it is not necessary to fit the model for all values of $k$, i.e., we can simply interpolate within first-order). Extensions to model selection using loss-functions defined on out-of-sample test data sets can also be derived, and can be carried over under appropriate assumptions to cross-validation. Note that the results described here do not have an obvious extension to non-linear models (such as feed-forward neural networks) or loss-functions such as the 0/1 loss for classification.

## Footnotes

[1]We assume the learning task consists of minimization of the loss function. If maximization is more appropriate, we can just consider minimization of the negative of the loss function.

## References

Bishop, C., *Neural Networks for Pattern Recognition*, Oxford University Press, 1995, pp. 376–377.

Cadez, I., D. Heckerman, C. Meek, P. Smyth, and S. White, 'Visualization of navigation patterns on a Web site using model-based clustering,' Technical Report MS-TR-00-18, Microsoft Research, Redmond, WA.

Li, Jonathan Q., and Barron, Andrew A., 'Mixture density estimation,' presented at NIPS 99.

Smyth, P., K. Ide, and M. Ghil, 'Multiple regimes in Northern hemisphere height fields via mixture model clustering,' *Journal of the Atmospheric Sciences*, vol. 56, no. 21, 3704–3723, 1999.
